# Discriminative Binaural Sound Localization

**Ehud Ben-Reuven** and **Yoram Singer**
School of Computer Science & Engineering
The Hebrew University, Jerusalem 91904, Israel
udi@benreuven.com, singer@cs.huji.ac.il

## Abstract

Time difference of arrival (TDOA) is commonly used to estimate the azimuth of a source in a microphone array. The most common methods to estimate TDOA are based on finding extrema in generalized cross-correlation waveforms. In this paper we apply microphone array techniques to a manikin head. By considering the entire cross-correlation waveform we achieve azimuth prediction accuracy that exceeds extrema locating methods. We do so by quantizing the azimuthal angle and treating the prediction problem as a multiclass categorization task. We demonstrate the merits of our approach by evaluating the various approaches on Sony's AIBO robot.

## 1 Introduction

In this paper we describe and evaluate several algorithms to perform sound localization in a commercial entertainment robot. The physical system being investigated is composed of a manikin head equipped with a two microphones and placed on a manikin body. This type of systems is commonly used to model sound localization in biological systems and the algorithms used to analyze the signal are usually inspired from neurology. In the case of an entertainment robot there is no need to be limited to a neurologically inspired model and we will use combination of techniques that are commonly used in microphone arrays and statistical learning. The focus of the work is the task of localizing an unknown stationary source (compact in location and broad in spectrum). The goal is to find the azimuth angle of the source relative to the head.

A common paradigm to approximately find the location of a sound source employs a microphone array and estimates time differences of arrival (TDOA) between microphones in the array (see for instance [1]). In a dual-microphone array it is usually assumed that the difference in the two channels is limited to a small time delay (or linear phase in frequency domain) and therefore the cross-correlation is peaked at the the time corresponding to the delay. Thus, methods that search for extrema in cross-correlation waveforms are commonly used [2]. The time delay approach is based on the assumption that the sound waves propagate along a single path from the source to the microphone and that the microphone response of the two channels for the given source location is approximately the same. In order for this to hold, the microphones should be identical, co-aligned, and, near each other relative to the source. In addition there should not be any obstructions between or near the microphones. The time delay assumption fails in the case of a manikin head: the microphone are antipodal and in addition the manikin head and body affect the response in a complex way. In our system the distance to the supporting floor was also significant. Our approach for overcoming these difficulties is composed of two stages. First, we perform signal processing based on the generalized cross correlation transform called Phase

Transform (PHAT) also called Cross Power Spectrum Phase (CPSP). This signal processing removes to a large extent variations due the sound source. Then, rather than proceeding with peak-finding we employ *discriminative* learning methods by casting the azimuth estimation as a *multiclass* prediction problem. The results achieved by combining the two stages gave improved results in our experimental setup.

This paper is organized as follows. In Sec. 2 we describe how the signal received in the two microphones was processed to generate accurate features. In Sec. 3 we outline the supervised learning algorithm we used. We then discuss in Sec. 4 approaches to combined predictions from multiple segments. We describe experimental results in Sec. 5 and conclude with a brief discussion in Sec. 6.

## 2  Signal Processing

Throughout the paper we denote signals in the time domain by lower case letters and in the frequency domain by upper case letters. We denote the convolution operator between two signals by $\star$ and the correlation operator by $\odot$. The unknown source signal is denoted by $s$ and thus its spectrum is $S$. The source signal passes through different physical setup and is received at the right and left microphones. We denote the received signals by $s^l$ and $s^r$. We model the different physical media, the signal passes through, as two linear systems whose frequency response is denoted by $H^r$ and $H^l$. In addition the signals are contaminated with noise that may account for non-linear effects such as room reverberations (see for instance [3] for more detailed noise models). Thus, the received signals can be written in the time and frequency domain as,

$$S^l = H^l S + \Xi_l \qquad s^l = h^l \star s + \epsilon_l \tag{1}$$

$$S^r = H^r S + \Xi_r \qquad s^r = h^r \star s + \epsilon_r \quad . \tag{2}$$

Since the source signal is typically non-stationary we break each training and test signal into segments and perform the processing described in the sequel based on short-time Fourier transform. Let $N$ be the number of segments a signal is divided into and $L$ the number of samples in a single segment. Each is multiplied by a Hanning window and padded with zeros to smooth the end-of-segment effects and increase the resolution of the short-time Fourier transform (see for instance [8]). Denote by $s_n^l$ and $s_n^r$ the left and right signal-segments after the above processing. Based on the properties of the Fourier transform, the local cross-correlation between the two signals can be computed efficiently by the inverse Fourier transform, denoted $\mathcal{F}^{-1}$, of the product of the spectrum of $s_n^l$ and the complex conjugate of the spectrum of $s_n^r$,

$$r_n = s_n^l \odot s_n^r = \mathcal{F}^{-1}\left(S_n^l \bar{S}_n^r\right) \quad . \tag{3}$$

Had the difference between the two signals been a mere time delay due to the different location of the microphones, the cross correlation would have obtained its maximal value at a point which corresponds to the time-lag between the received signals. However, since the source signal passes through different physical media the short-time cross-correlation does not necessarily obtain a large value at the time-lag index. It is therefore common (see for instance [1]) to multiply the spectrum of the cross-correlation by a weighting function in order to compensate for the differences in the frequency responses obtained at the two microphones. Denoting the spectral shaping function for the $n$th segment by $\psi_n$, the generalization cross-correlation from Eq. (3) is, $r_n = s_n^l \odot s_n^r = \mathcal{F}^{-1}\left(\psi_n S_n^l \bar{S}_n^r\right)$. For "plain" cross-correlation, $\psi_n(j)$ is equal to 1 at each (discrete) frequency $j$. In our tests we found that a globally-equalized cross-correlation gives better results. The transform is obtained by setting, $\psi_n(j) = 1/e_j$ where $e_j$ is the average over all measurements and both channels of $|S(j)|^2$. Finally, for PHAT the weight for the spectral point $j$ is,

$$\psi_n(j) = \frac{1}{|S_n^l(j)\bar{S}_n^r(j)|} \quad .$$

To further motivate and explain the PHAT weighting scheme, we build on the derivation in [5] and expand the PHAT assuming that the noise is zero. In PHAT the spectral value at frequency point $j$ (prior to the inverse Fourier transform) is,

$$\psi_n(j)\, S_n^l(j)\, \bar{S}_n^r(j) = \frac{S_n^l(j)\bar{S}_n^r(j)}{|S_n^l(j)\bar{S}_n^r(j)|} \quad . \tag{4}$$

Inserting Eq. (1) and Eq. (2) into Eq. (4) without noise we get,

$$\psi_n(j)\, S_n^l(j)\, \bar{S}_n^r(j) = \frac{H_n^l(j)S_n(j)\bar{H}_n^r(j)\bar{S}_n(j)}{|H_n^l(j)S_n(j)\ \bar{H}_n^r(j)\bar{S}_n(j)|} = \frac{H^l\bar{H}^r}{|H^l||H^r|} \quad . \tag{5}$$

Therefore, assuming the noise is zero, PHAT eliminates the contribution of the unknown source $S$ and the entire waveform of PHAT is only a function of the physical setup. If all other physical parameters are constant, the PHAT waveform (as well as its peak location) is a function of the azimuth angle $\theta$ of the sound source relative to the manikin head. This is of course an approximation and the presence of noise and changes in the environment result in a waveform that deviates from the closed-form given in Eq. (5). In Fig. 1 we show the empirical average of the waveform for PHAT and for the equalized cross-correlation, the vertical bars represent an error of $1\sigma$. In both cases, the location of the maximal correlation is clearly at $0$ as expected. Nonetheless, the high variance, especially in the case of the equalized cross-correlation imply that classification of individual segments may often be rather difficult.

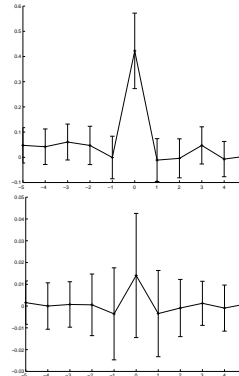

Figure 1: Average waveform with standard deviation for $\theta = 0^o$ after performing PHAT (top) and the equalized cross-correlation (bottom).

In practice, we found that it suffices to take only the energetic portion of the generalized cross-correlation waveforms by considering only time lags of $-l$ through $l$ samples. In what follows we will take this part to be the waveform. Formally, the feature vector of the $n$th segment is defined as,

$$\mathbf{x}_n = (r_n(-l), \ldots, r_n(l)) \tag{6}$$

were $l$ was set to be bigger than the maximal lag in samples between the two channels, $l \geq D/c$ where $D$ is the head diameter and $c$ is speed of sound.

Summarising, the signal processing we perform is based on short time Fourier transform of the signals received at the two microphones. From the two spectrums we then compute the generalized cross-correlation using one of the three weighting schemes described above and taking only $2l + 1$ samples of the resulting waveforms as the feature vectors. We now move our focus to classification of a single segment.

## 3 Single Segment Classification

Traditional approaches to sound localization search for the the position of the extreme value in the generalized cross-correlation waveform that were derived in Sec. 2. While being intuitive, this approach is prone to noise. Peak location can be considered as a reduction in dimensionality, from $2l + 1$ to $1$, of the feature vectors $\mathbf{x}_n$, however we have shown in Eq. 5 that the entire waveform of PHAT can be used as a feature vector to localise the source. Indeed, in Sec. 5 we report experimental results which show that peak-finding is significantly inferior to methods that we now describe, that uses the entire waveform. In all techniques, peak-location and waveform, we used supervised learning to build a model of the data using a training set and then used a test set to evaluate the learned model.

In a supervised learning setting, we have access to *labelled* examples and the goal is to find a mapping from the instance domain (the peak-location or waveforms in our setting) to a response variable (the azimuth angle). Since the angle is a continuous variable the first approach that comes to mind is using a linear or non-linear regressors. However, we found that regression algorithms such as Widrow-Hoff [10] yielded inferior results. Instead of treating the learning problem as a regression problem, we quantized the angle and converted the sound localization problem into a multiclass decision problem. Formally, we bisected the interval $[-90^o, +90^o]$ into $K$ non-overlapping intervals $\theta^{(1)}, \ldots, \theta^{(K)}$ where $\theta^{(i)} = [\theta^i - \Delta\theta/2, \theta^i + \Delta\theta/2)$ and $\Delta\theta = 180^o/(K-1)$. We now can transform the real-valued angle of the $n$th segment, $\theta_n$, into a discrete variable $y_n \in \{1, \ldots, K\}$ where $y_n = k$ iff $\theta_n \in \theta^{(k)}$. After this quantization, the training set is composed of instance-label pairs $\{x_n, y_n\}_{n=1}^N$ and the first task is to find a classification rule from the peak-location or waveforms space into $\{1, \ldots, K\}$. We will first describe the method used for peak-location and then we will describe two *discriminative* methods to classify the waveform. The first is based on a multiclass version of the Fisher linear discriminant [7] and is very simple to implement. The second employs recent advances in statistical learning and can be used in an online fashion. It thus can cope, to some extent with changes, in the environment such as moving elements that change the reverberation properties of the physical media.

**Peak location classification:**   Due to the relative low sampling frequency ($F_s = 16kHz$) spline interpolation was used to improve the peak location. In microphone arrays it is common to translate the peak-location to an estimate of the source azimuth using a geometric formula. However, this was found to be inappropriate due to the internal reverberations generated by the manikin head. We thus used the classification method describe in [4].

The peak location data was modelled using a separate histogram for each direction $k$. For a given direction $\theta^{(k)}$, all the training measurements $\mathbf{x}_n$ for which $\theta_n = \theta^{(k)}$ are used to build a single histogram: $H(r|k) = \sum_{n:y_n=k} [\![ \mathbf{x}_n \in [r\Delta, (r+1)\Delta) ]\!]$ where $[\![ \pi ]\!]$ is 1 if $\pi$ is true and 0 otherwise, $\Delta$ is the size of the bin in the histogram, $\Delta = \frac{2l}{|H|-1}$, and $|H|$ is the number of bins. An estimate of the probability density function was taken to be the normalized histogram step function: $\hat{P}(\mathbf{x}|k) = H(r : \mathbf{x} \in [r\Delta, (r+1)\Delta)|k)/N_k$ where $N_k$ is the number of training measurements for which $y_n = k$.

In order to classify new test data we simply compute the likelihood of the observed measurement under each distribution and choose the class attaining the maximal likelihood (ML) score with respect to the distribution defined by the histogram,

$$\hat{y} = \arg\max_k \hat{P}(\mathbf{x}|k) \ . \tag{7}$$

**Multiclass Fisher discriminant:**   Generalising the Fisher discriminant for binary classification problems to multiclass settings, each class is modelled as a multivariate normal distribution. To do so we divide the training set into subsets where the $k$th subset corresponds to measurements from azimuth in $\theta^{(k)}$. The density function of the $k$th class is

$$P(\mathbf{x}|k) = \frac{1}{\sqrt{(2\pi)^n |\mathbf{C}_k|}} \exp\left(-\frac{1}{2}(\mathbf{x} - \mu_k)^T \mathbf{C}_k^{-1} (\mathbf{x} - \mu_k)\right) \ ,$$

where $\mathbf{x}^T$ is the transpose of $\mathbf{x}$, $n = 2l+1$ is its dimensionality, $\mu_k$ denotes the mean of the normal distribution, and $\mathbf{C}_k$ the covariance matrix. Each mean and covariance matrix are set to be the maximum likelihood estimates,

$$\hat{\mu}_k = \frac{1}{N_k} \sum_{n:y_n=k} \mathbf{x}_n \ ; \ \hat{\mathbf{C}}_k = \frac{1}{N_k-1} \sum_{n:y_n=k} (\mathbf{x}_n - \hat{\mu}_k)^T (\mathbf{x}_n - \hat{\mu}_k) \ .$$

New test waveforms were then classified using the ML formula, Eq. 7.

The advantage of Fisher linear discriminant is that it is simple and easy to implement. However, it degenerates if the training data is non-stationary, as often is the case in sound localization problems due to effects such as moving objects. We therefore also designed, implemented and tested a second discriminative methods based on the Perceptron.

**Online Learning using Multiclass Perceptron with Kernels:** Despite, or because of, its age the Perceptron algorithm [9] is a simple and effective algorithm for classification. We chose the Perceptron algorithm for its simplicity, adaptability, and ease in incorporating Mercer kernels described below. The Perceptron algorithm is a conservative online algorithm: it receives an instance, outputs a prediction for the instance, and only in case it made a prediction mistake the Perceptron update its classification rule which is a hyperplane. Since our setting requires building a multiclass rule, we use the version described in [6] which generalises the Perceptron to multiclass settings. We first describe the general form of the algorithm and then discuss the modifications we performed in order to adapt it to the sound localization problem.

To extend the Perceptron algorithm to multiclass problem we maintain $K$ hyperplanes (one per class) denoted $\mathbf{w}_1, \ldots, \mathbf{w}_K$. The algorithm works in an online fashion working on one example at a time. On the $n$th round, the algorithm gets a new instance $\mathbf{x}_n$ and set the predicted class to be the index of the hyperplane attaining the largest inner-product with the input instance, $\hat{y}_n = \arg\max_k \ \mathbf{w}_k \cdot \mathbf{x}_n$ . If the algorithm made a prediction error, that is $\hat{y}_n \neq y_n$, it updates the set of hyperplanes. In [6] a family of possible update schemes was given. In this work we have used the so called *uniform* update which is very simple to implement and also attained very good results. The uniform update moves the hyperplane corresponding to the correct label $\mathbf{w}_{y_n}$ in the direction of $\mathbf{x}_n$ and all the hyperplanes whose inner-products were larger than $\mathbf{w}_{y_n} \cdot \mathbf{x}_n$ away from $\mathbf{x}_n$. Formally, let $\Delta_n = \{k \mid k \neq y_n \, ; \, \mathbf{w}_k \cdot \mathbf{x}_n \geq \mathbf{w}_{y_n} \cdot \mathbf{x}_n\}$ . We update the hyperplanes as follows,

$$\mathbf{w}_k = \mathbf{w}_k + \begin{cases} \mathbf{x}_n & k = y_n \\ -\frac{1}{|\Delta_n|}\mathbf{x}_n & k \in \Delta_n \end{cases} , \tag{8}$$

and if $k \notin \Delta_n \cup \{y_n\}$ then we keep $\mathbf{w}_k$ intact. This update of the hyperplanes is performed only on rounds on which there was a prediction error. Furthermore, on such rounds only a subset of the vectors is updated and thus the algorithm is called ultraconservative. The multiclass Perceptron algorithm is guaranteed to converge to a perfect classification rule if the data can be classified perfectly by an unknown set of hyperplanes. When the data cannot be classified perfectly then an alternative competitive analysis can be applied.

The problem with above algorithm is that it allows only linear classification rules. However, linear classifiers may not suffice to obtain in many applications, including the sound localization application. We therefore incorporate kernels into the multiclass Perceptron. A kernel is an inner-product operator $K : X \times X \to \mathbb{R}$ where $X$ is the instance space (for instance, PHAT waveforms). An explicit way to describe $K$ is via a mapping $\phi : X \to \tilde{X}$ from $X$ to an inner-products space $\tilde{X}$ such that $K(x, x') = \phi(x) \cdot \phi(x')$. Common kernels are RBF kernels and polynomial kernels which take the form $K(\mathbf{x}, \mathbf{x}') = (a + \mathbf{x} \cdot \mathbf{x}')^d$. Any learning algorithm that is based on inner-products with a weighted sum of vectors can be converted to a kernel-based version by explicitly keeping the weighted combination of vectors. In the case of the multiclass Perceptron we replace the update from Eq. 8 with a "kernelized" version,

$$\mathbf{w}_k = \mathbf{w}_k + \begin{cases} \phi(\mathbf{x}_n) & k = y_n \\ -\frac{1}{|\Delta_n|}\phi(\mathbf{x}_n) & k \in \Delta_n \end{cases} . \tag{9}$$

Since we cannot compute $\phi(\mathbf{x}_n)$ explicitly we instead perform bookkeeping of the weights associated with each $\phi(\mathbf{x}_n)$ and compute a inner-products using the kernel functions. For instance, the inner-product of a vector $\mathbf{w} = \sum_i \alpha_i \phi(\mathbf{x}_i)$ with a new instance $\mathbf{x}'$ is $\mathbf{w} \cdot \mathbf{x}' = \sum_i \alpha_i \phi(\mathbf{x}_i) \cdot \phi(\mathbf{x}') = \sum_i \alpha_i K(\mathbf{x}_i, \mathbf{x}')$.

| Algorithm | Err | $\ell_1$ |
|---|---|---|
| PHAT + Poly Kernels, D=5 | $37.6\% \pm 0.2\%$ | $20^o \pm 0.2^o$ |
| PHAT + Fisher | $37.8\% \pm 0.2\%$ | $20.2^o \pm 0.2^o$ |
| PHAT + Peak-finding | $44.8\% \pm 0.2\%$ | $25.8^o \pm 0.2^o$ |
| Equalized CrossCor + Peak-finding | $59.1\% \pm 0.1\%$ | $35.7^o \pm 0.2^o$ |

Table 1: Summary of results of sound localization methods for a single segment.

In our experiments we found that polynomial kernel of degree 5 yielded the best results. The results are summarised in Table 1. We defer the discussion of the results to Sec. 5.

## 4 Multi-segment Classification

The accuracy of a single segment classifier is too low to make our approach practical. However, if the source of sound does not move for a period of time, we can accumulate evidence from multiple segments in order to increase the accuracy. Due to the lack of space we only outline the multi-segment classification procedure for the Fisher discriminant and compare it to smoothing and averaging techniques used in the signal processing community.

In multi-segment classification we are given $f$ waveforms for which we assume that the source angle did not change in this period, i.e., $\theta_n^j = \theta_n$, $\forall j = 1, \ldots, f$. Each small window was processed independently to give a feature vector $\mathbf{x}_n^j$. We then converted the waveform feature vector into a probability estimate for each discrete angle direction, $P(\mathbf{x}_n^j | \theta^{(k)})$ using the Fisher discriminant. We next assumed that the probability estimates for consecutive windows are independent. This is of course a false assumption. However, we found that methods which compensate for the dependencies did not yield substantial improvements. The probability density function of the entire window is therefore $\hat{P}(\mathbf{x}_n^{1\cdots f} | \theta^{(k)}) = \prod_{j=1}^{f} \hat{P}(\mathbf{x}_n^j | \theta^{(k)})$ , and the ML estimation for $\hat{\theta}_n$ is $\hat{\theta}_n = arg\max_{\theta^{(k)}} \hat{P}(\mathbf{x}_n^{1\cdots f} | \theta^{(k)})$ . We compared the Maximum Likelihood decision under the independence assumption with the following commonly used signal processing technique. We averaged the power spectrum and cross power spectrum of the different windows and only then we proceeded to compute the generalized cross correlation waveform, $r_n = \mathcal{F}^{-1}\left(\psi_n E\left\{S_n^l \bar{S}_n^r\right\}\right)$ , where $E\{\cdot\}$ is the average over the measurements in the same window, $E\{Z_n\} = \frac{1}{f}\sum_{j=1}^{f} Z_n^j$ . The averaged weight function for the PHAT waveform is now $\psi_n(j) = 1/|E\left\{S_n^l(j)\bar{S}_n^r(j)\right\}|$ . When using averaged power spectrum it is also possible to define a smoothed coherent transform (SCOT) [1]. The weight vector in this case is identical to the PHAT weight in the single segment case, $\psi_n(j) = 1/\sqrt{E\left\{S_n^l(j)\bar{S}_n^l(j)\right\} E\left\{S_n^r(j)\bar{S}_n^r(j)\right\}}$. Finally, we applied the classification techniques for the single segments on the resulting (smoothed or averaged) waveform.

## 5 Experimental Results

In this section we report and discuss results of experiments that we performed with the various learning algorithms for single-segments and multiple segments. Measurements where made using the Sony ERS-210 AIBO robot. The sampling frequency was fixed to $F_s = 16kHz$ and the robot's uni-directional microphone without automatic level control was used. The robot was laid on a concrete floor in a regular office room, the room reverberations was $T_{60} \approx 0.6sec$. A loudspeaker, playing speech data from multiple speakers, was placed $190cm$ in front of the robot and $90cm$ above its plane, the background noise was $SNR \approx 40db$. A PC connected through a wireless link to the robot directed its head relative to the speaker. The location of the sound source was limited to be in front of the head ($\theta = -90^o \ldots + 90^o$) at a fixed constant elevation and in jumps of $+10^o$. Therefore, the number of classes, $K$, for training is 19. An illustration of the system is given in Fig. 2.

| Algorithm | Err | $\ell_1$ |
|---|---|---|
| Max. Likl. PHAT + Fisher | $3.7\% \pm 0.4\%$ | $0.4^{\circ} \pm 0.1^{\circ}$ |
| SCOT + Fisher | $4.8\% \pm 0.4\%$ | $0.8^{\circ} \pm 0.2^{\circ}$ |
| Smoothed PHAT + Fisher | $6.4\% \pm 0.7\%$ | $1.3^{\circ} \pm 0.2^{\circ}$ |
| Smoothed PHAT + Peak-finding | $7.6\% \pm 0.7\%$ | $1.3^{\circ} \pm 0.3^{\circ}$ |
| SCOT + Peak-finding | $7.8\% \pm 0.7\%$ | $1.4^{\circ} \pm 0.3^{\circ}$ |

Table 2: Summary of results of sound localization methods for multiple segments.

Further technical details can be obtained from http://udi.benreuven.com. (MATLAB is a trademark of Mathworks, Inc. and AIBO is a trademark of Sony and its affiliates.) For each head direction 4000 segments of data were collected. Each segment is $16msec$ long. The segments were collected with a partial overlap of $10msec$. For each direction, the measurements were divided into equal amounts of train and test measurements. The total number of segments per class, $N_k$, is 2000. Therefore, altogether there were $N = N_k \times K = 38,000$ segments for training and the same amount for evaluation. An FFT of size $512$ was used to generate un-normalized cross-correlations, equalized cross-correlations, and PHAT waveforms. From the transformed waveforms 11 samples where taken ($l = 5$ in Eq. 6). Extrema locations in histograms were found using $|H| = 41$ bins.

We used two evaluation measures for comparing the different algorithms. The first, denoted $Err$, is the empirical classification error that counts the number of times the predicted (discretized) angle was different than the true angle, that is, $Err = \frac{1}{N} \sum_{n=1}^{N} [\![ \theta_n \neq \hat{\theta}_n ]\!]$. The second evaluation measure, denoted $\ell_1$, is the average absolute difference between the predicted angle and the true angle, $\ell_1 = \frac{1}{N} \sum_{n=1}^{N} |\hat{\theta}_n - \theta_n|$. It should be kept in mind that the test data was obtained from the same direction set as the training data. Therefore, $\ell_1$ is an appropriate evaluation measure of the errors in our experimental setting. However, alternative evaluation methods should be devised for general recordings when the test signal is not confined to a finite set of possible directions.

The accuracy results with respect to both measures on the test data for the various representations and algorithms are summarized in Table 1. It is clear from the results that traditional methods which search for extrema in the waveforms are inferior to the discriminative methods. As a by-product we confirmed that equalized cross-correlations is inferior to PHAT modelling for high SNR with strong reverberations, similar results were reported in [11]. The two discriminative methods achieve about the same results. Using the Perceptron algorithm with degree 5

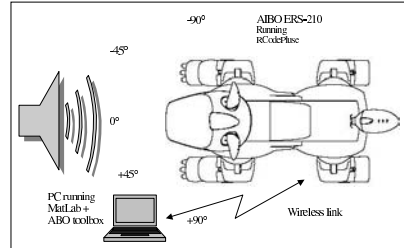

Figure 2: Acquisition system overview.

achieves the best results but the difference between the Perceptron and the multiclass Fisher discriminant is not statistically significant. It is worth noting again that we also tested linear regression algorithms. Their performance turns to be inferior to the discriminative multiclass approaches. A possible explanation is that the multiclass methods employ *multiple* hyperplanes and project each class onto a different hyperplane while linear regression methods seek a *single* hyperplane onto which example are projected.

Although Fisher's discriminant and the Perceptron algorithm exhibit practically the same performance, they have different merits. While Fisher's discriminant is very simple to implement and is space efficient the Perceptron is capable to adapt quickly and achieves high accuracy even with small amounts of training data. In Fig 3 we compare the error rates of Fisher's discriminant and the Perceptron on subsets of the training data. The Perceptron clearly outperforms Fisher's discriminant when the number of training examples is less than 3000 but once about 5000 examples are pro-

vided the two algorithms are indistinguishable. This suggests that online algorithms may be more suitable when the sound source is stationary only for short periods.

Last we compared multi-segment results. Multi-segment classification was performed by taking $f = 41$ consecutive measurements over a window of $256msec$ during which the source location remained fix. In Table 2 we report classification results for the various multi-segment techniques. (Since the Perceptron algorithm used a very large number of kernels we did not implement a multi-segment classification using the Perceptron. We are currently conducting research on space-efficient kernel-based methods for multi-segment classification.) Here again, the best performing method is Fisher's discriminant that combines the scores directly without averaging and smoothing leads the pack. The resulting prediction accuracy of Fisher's discriminant is good enough to make the solution prac-

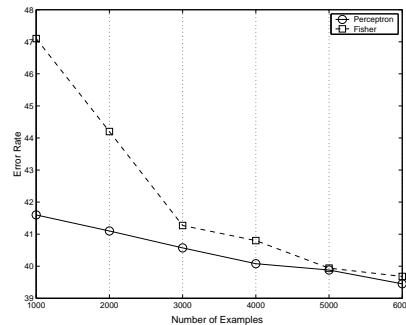

Figure 3: Error rates of Fisher's discriminant and the Perceptron for various training sizes.

tical so long as the sound source is fixed and the recording conditions do not change.

## 6 Discussion

We have demonstrated that by using discriminative methods highly accurate sound localization is achievable on a small commercial robot equipped with a binaural hearing that are placed inside a manikin head. We have confirmed that PHAT is superior to plain cross-correlation. For classification using multiple segments classifying the entire PHAT waveform gave better results than various techniques that smooth the power spectrum over the segments. Our current research is focused on efficient discriminative methods for sound localization in changing environments.

## References

[1] C. H. Knapp and G. C. Carter. The generalized correlation method for estimation of time delay. IEEE Transactions on ASSP, 24(4):320-327,1976.

[2] M. Omologo and P. Svaizer. Acoustic event localization using a crosspowerspectrum phase based technique. Proceedings of ICASSP1994, Adelaide, Australia, 1994.

[3] T. Gustafsson and B.D. Rao. Source Localization in Reverberant Environments: Statistical Analysis. Submitted to IEEE Trans. on Speech and Audio Processing, 2000.

[4] N. Strobel and R. Rabenstein. Classification of Time Delay Estimates for Robust Speaker Localization ICASSP, Phoenix, USA, March 1999.

[5] J. Benesty Adaptive eigenvalue decomposition algorithm for passive acoustic source localization J. Acoust. Soc. Am. 107 (1), January 2000

[6] K. Crammer and Y. Singer. Ultraconservative online algorithms for multiclass problems. In *Proc. of the 14th Annual Conf. on Computational Learning Theory*, 2001.

[7] R. O. Duda, P. E. Hart. *Pattern Classification*. Wiley, 1973.

[8] B. Porat. *A course in Digital Signal Processing*. Wiley, 1997.

[9] F. Rosenblatt. The Perceptron: A probabilistic model for information storage and organization in the brain. *Psychological Review*, 65:386–407, 1958.

[10] B. Widrow and M. E. Hoff. Adaptive switching circuits. *1960 IRE WESCON Convention Record*, pages 96–104, 1960.

[11] P. Aarabi, A. Mahdavi. The Relation Between Speech Segment Selectivity and Time-Delay Estimation Accuracy. In *Proc. of IEEE Conf. on Acoustics Speech and Signal Processing*, 2002.
